# Offline Handwriting Recognition with Multidimensional Recurrent Neural Networks

**Alex Graves**
TU Munich, Germany
graves@in.tum.de

**Jürgen Schmidhuber**
IDSIA, Switzerland and TU Munich, Germany
juergen@idsia.ch

## Abstract

Offline handwriting recognition—the automatic transcription of images of handwritten text—is a challenging task that combines computer vision with sequence learning. In most systems the two elements are handled separately, with sophisticated preprocessing techniques used to extract the image features and sequential models such as HMMs used to provide the transcriptions. By combining two recent innovations in neural networks—multidimensional recurrent neural networks and connectionist temporal classification—this paper introduces a globally trained offline handwriting recogniser that takes raw pixel data as input. Unlike competing systems, it does not require any alphabet specific preprocessing, and can therefore be used unchanged for any language. Evidence of its generality and power is provided by data from a recent international Arabic recognition competition, where it outperformed all entries (91.4% accuracy compared to 87.2% for the competition winner) despite the fact that neither author understands a word of Arabic.

## 1   Introduction

Offline handwriting recognition is generally observed to be harder than online handwriting recognition [14]. In the online case, features can be extracted from both the pen trajectory and the resulting image, whereas in the offline case only the image is available. Nonetheless, the standard recognition process is essentially the same: a sequence of features are extracted from the data, then matched to a sequence of labels (usually characters or sub-character strokes) using either a hidden Markov model (HMM) [9] or an HMM-neural network hybrid [10].

The main drawback of this approach is that the input features must meet the stringent independence assumptions imposed by HMMs (these assumptions are somewhat relaxed in the case of hybrid systems, but long-range input dependencies are still problematic). In practice this means the features must be redesigned for every alphabet, and, to a lesser extent, for every language. For example it would be impossible to use the same system to recognise both English and Arabic.

Following our recent success in transcribing raw online handwriting data with recurrent networks [6], we wanted to build an offline recognition system that would work on raw pixels. As well as being alphabet-independent, such a system would have the advantage of being globally trainable, with the image features optimised along with the classifier.

The online case was relatively straightforward, since the input data formed a 1D sequence that could be fed directly to a recurrent network. The *long short-term memory* (LSTM) network architecture [8, 3] was chosen for its ability to access long-range context, and the *connectionist temporal classification* [5] output layer allowed the network to transcribe the data with no prior segmentation.

The offline case, however, is more challenging, since the input is no longer one-dimensional. A naive approach would be to present the images to the network one vertical line at a time, thereby transforming them into 1D sequences. However such a system would be unable to handle distor-

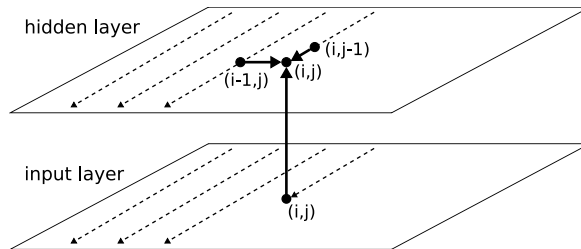

Figure 1: **Two dimensional MDRNN**. The thick lines show connections to the current point $(i, j)$. The connections within the hidden layer plane are recurrent. The dashed lines show the scanning strips along which previous points were visited, starting at the top left corner.

tions along the vertical axis; for example the same image shifted up by one pixel would appear completely different. A more flexible solution is offered by *multidimensional recurrent neural networks* (MDRNNs) [7]. MDRNNs, which are a special case of directed acyclic graph networks [1], generalise standard RNNs by providing recurrent connections along all spatio-temporal dimensions present in the data. These connections make MDRNNs robust to local distortions along any combination of input dimensions (e.g. image rotations and shears, which mix vertical and horizontal displacements) and allow them to model multidimensional context in a flexible way. We use multidimensional LSTM because it is able to access long-range context.

The problem remains, though, of how to transform two-dimensional images into one-dimensional label sequences. Our solution is to pass the data through a hierarchy of MDRNN layers, with blocks of activations gathered together after each level. The heights of the blocks are chosen to incrementally collapse the 2D images onto 1D sequences, which can then be labelled by the output layer. Such hierarchical structures are common in computer vision [15], because they allow complex features to be built up in stages. In particular our multilayered structure is similar to that used by convolution networks [11], although it should be noted that because convolution networks are not recurrent, they cannot be used for cursive handwriting recognition without presegmented inputs.

The method is described in detail in Section 2, experimental results are given in Section 3, and conclusions and directions for future work are given in Section 4.

## 2   Method

The three components of our recognition system are: (1) multidimensional recurrent neural networks, and multidimensional LSTM in particular; (2) the connectionist temporal classification output layer; and (3) the hierarchical structure. In what follows we describe each component in turn, then show how they fit together to form a complete system. For a more detailed description of (1) and (2) we refer the reader to [4]

### 2.1   Multidimensional Recurrent Neural Networks

The basic idea of multidimensional recurrent neural networks (MDRNNs) [7] is to replace the single recurrent connection found in standard recurrent networks with as many connections as there are spatio-temporal dimensions in the data. These connections allow the network to create a flexible internal representation of surrounding context, which is robust to localised distortions.

An MDRNN hidden layer scans through the input in 1D strips, storing its activations in a buffer. The strips are ordered in such a way that at every point the layer has already visited the points one step back along every dimension. The hidden activations at these previous points are fed to the current point through recurrent connections, along with the input. The 2D case is illustrated in Fig. 1.

One such layer is sufficient to give the network access to all context against the direction of scanning from the current point (e.g. to the top and left of $(i, j)$ in Fig. 1). However we usually want surrounding context in all directions. The same problem exists in 1D networks, where it is often useful to have information about the future as well as the past. The canonical 1D solution is bidi-

rectional recurrent networks [16], where two separate hidden layers scan through the input forwards and backwards. The generalisation of bidirectional networks to $n$ dimensions requires $2^n$ hidden layers, starting in every corner of the $n$ dimensional hypercube and scanning in opposite directions. For example, a 2D network has four layers, one starting in the top left and scanning down and right, one starting in the bottom left and scanning up and right, etc. All the hidden layers are connected to a single output layer, which therefore receives information about all surrounding context.

The error gradient of an MDRNN can be calculated with an n-dimensional extension of backpropagation through time. As in the 1D case, the data is processed in the reverse order of the forward pass, with each hidden layer receiving both the output derivatives and its own $n$ 'future' derivatives at every timestep.

Let $a_j^{\mathbf{P}}$ and $b_j^{\mathbf{P}}$ be respectively the input and activation of unit $j$ at point $\mathbf{p} = (p_1, \ldots, p_n)$ in an n-dimensional input sequence $\mathbf{x}$ with dimensions $(D_1, \ldots, D_n)$. Let $\mathbf{p}_d^- = (p_1, \ldots, p_d - 1, \ldots, p_n)$ and $\mathbf{p}_d^+ = (p_1, \ldots, p_d + 1, \ldots, p_n)$. Let $w_{ij}$ and $w_{ij}^d$ be respectively the weight of the feedforward connection from unit $i$ to unit $j$ and the recurrent connection from $i$ to $j$ along dimension $d$. Let $\theta_h$ be the activation function of hidden unit $h$, and for some unit $j$ and some differentiable objective function $O$ let $\delta_j^{\mathbf{P}} = \frac{\partial O}{\partial a_j^{\mathbf{P}}}$. Then the forward and backward equations for an n-dimensional MDRNN with $I$ input units, $K$ output units, and $H$ hidden summation units are as follows:

**Forward Pass**

$$a_h^{\mathbf{P}} = \sum_{i=1}^{I} x_i^{\mathbf{P}} w_{ih} + \sum_{\substack{d=1: \\ p_d > 0}}^{n} \sum_{\hat{h}=1}^{H} b_{\hat{h}}^{\mathbf{P}_d^-} w_{\hat{h}h}^d$$

$$b_h^{\mathbf{P}} = \theta_h(a_h^{\mathbf{P}})$$

**Backward Pass**

$$\delta_h^{\mathbf{P}} = \theta_h'(a_h^{\mathbf{P}}) \left( \sum_{k=1}^{K} \delta_k^{\mathbf{P}} w_{hk} + \sum_{\substack{d=1: \\ p_d < D_d - 1}}^{n} \sum_{\hat{h}=1}^{H} \delta_{\hat{h}}^{\mathbf{P}_d^+} w_{h\hat{h}}^d \right)$$

### 2.1.1 Multidimensional LSTM

Long Short-Term Memory (LSTM) [8, 3] is an RNN architecture designed for data with long-range interdependencies. An LSTM layer consists of recurrently connected 'memory cells', whose activations are controlled by three multiplicative gate units: the input gate, forget gate and output gate. The gates allows the cells to store and retrieve information over time, giving them access to long-range context.

The standard formulation of LSTM is explicitly one-dimensional, since each cell contains a single recurrent connection, whose activation is controlled by a single forget gate. However we can extend this to $n$ dimensions by using instead $n$ recurrent connections (one for each of the cell's previous states along every dimension) with $n$ forget gates.

Consider an MDLSTM memory cell in a hidden layer of $H$ cells, connected to $I$ input units and $K$ output units. The subscripts $c$, $\iota$, $\phi$ and $\omega$ refer to the cell, input gate, forget gate and output gate respectively. $b_h^{\mathbf{P}}$ is the output of cell $h$ in the hidden layer at point $\mathbf{p}$ in the input sequence, and $s_c^{\mathbf{P}}$ is the *state* of cell $c$ at $\mathbf{p}$. $f_1$ is the activation function of the gates, and $f_2$ and $f_3$ are respectively the cell input and output activation functions. The suffix $\phi, d$ denotes the forget gate corresponding to recurrent connection $d$. The input gate $\iota$ is connected to previous cell $c$ along all dimensions with the same weight $(w_{c\iota})$ whereas the forget gates are connected to cell $c$ with a separate weight $w_{c(\phi,d)}$ for each dimension $d$. Then the forward and backward equations are as follows:

**Forward Pass**

$$\text{\underline{\textit{Input Gate:}}} \quad b_\iota^{\mathbf{P}} = f_1 \left( \sum_{i=1}^{I} x_i^{\mathbf{P}} w_{i\iota} + \sum_{\substack{d=1: \\ p_d > 0}}^{n} \left( w_{c\iota} s_c^{\mathbf{P}_d^-} + \sum_{h=1}^{H} b_h^{\mathbf{P}_d^-} w_{h\iota}^d \right) \right)$$

$$\text{\underline{\textit{Forget Gate:}}} \quad b_{\phi,d}^{\mathbf{P}} = f_1 \left( \sum_{i=1}^{I} x_i^{\mathbf{P}} w_{i(\phi,d)} + \sum_{\substack{d'=1: \\ p_{d'} > 0}}^{n} \sum_{h=1}^{H} b_h^{\mathbf{P}_{d'}^-} w_{h(\phi,d)}^{d'} + \begin{cases} w_{c(\phi,d)} s_c^{\mathbf{P}_d^-} & \text{if } p_d > 0 \\ 0 & \text{otherwise} \end{cases} \right)$$

$$\underline{\textit{Cell:}} \quad a_c^{\mathbf{P}} = \sum_{i=1}^{I} x_i^{\mathbf{P}} w_{ic} + \sum_{\substack{d=1:\\ p_d>0}}^{n} \sum_{h=1}^{H} b_h^{\mathbf{P}_d^{\overline{}}} w_{hc}^{d} \qquad \underline{\textit{State:}} \quad s_c^{\mathbf{P}} = b_\iota^{\mathbf{P}} f_2(a_c^{\mathbf{P}}) + \sum_{\substack{d=1:\\ p_d>0}}^{n} s_c^{\mathbf{P}_d^{\overline{}}} b_{\phi,d}^{\mathbf{P}}$$

$$\underline{\textit{Output Gate:}} \quad b_\omega^{\mathbf{P}} = f_1 \left( \sum_{i=1}^{I} x_i^{\mathbf{P}} w_{i\omega} + \sum_{\substack{d=1:\\ p_d>0}}^{n} \sum_{h=1}^{H} b_h^{\mathbf{P}_d^{\overline{}}} w_{h\omega}^{d} + w_{c\omega} s_c^{\mathbf{P}} \right)$$

$$\underline{\textit{Cell Output:}} \quad b_c^{\mathbf{P}} = b_\omega^{\mathbf{P}} f_3(s_c^{\mathbf{P}})$$

**Backward Pass**

$$\underline{\textit{Cell Output:}} \quad \epsilon_c^{\mathbf{P}} \stackrel{\text{def}}{=} \frac{\partial O}{\partial b_c^{\mathbf{P}}} = \sum_{k=1}^{K} \delta_k^{\mathbf{P}} w_{ck} + \sum_{\substack{d=1:\\ p_d<D_d-1}}^{n} \sum_{h=1}^{H} \delta_h^{\mathbf{P}_d^{+}} w_{ch}^{d}$$

$$\underline{\textit{Output Gate:}} \quad \delta_\omega^{\mathbf{P}} = f_1'(a_\omega^{\mathbf{P}}) \epsilon_c^{\mathbf{P}} f_3(s_c^{\mathbf{P}})$$

$$\underline{\textit{State:}} \quad \epsilon_s^{\mathbf{P}} \stackrel{\text{def}}{=} \frac{\partial O}{\partial s_c^{\mathbf{P}}} = b_\omega^{\mathbf{P}} f_3'(s_c^{\mathbf{P}}) \epsilon_c^{\mathbf{P}} + \delta_\omega^{\mathbf{P}} w_{c\omega} + \sum_{\substack{d=1:\\ p_d<D_d-1}}^{n} \left( \epsilon_s^{\mathbf{P}_d^{+}} b_{\phi,d}^{\mathbf{P}_d^{+}} + \delta_\iota^{\mathbf{P}_d^{+}} w_{c\iota} + \delta_{\phi,d}^{\mathbf{P}_d^{+}} w_{c(\phi,d)} \right)$$

$$\underline{\textit{Cell:}} \quad \delta_c^{\mathbf{P}} = b_\iota^{\mathbf{P}} f_2'(a_c^{\mathbf{P}}) \epsilon_s^{\mathbf{P}} \qquad \underline{\textit{Forget Gate:}} \quad \delta_{\phi,d}^{\mathbf{P}} = \begin{cases} f_1'(a_{\phi,d}^{\mathbf{P}}) s_c^{\mathbf{P}_d^{\overline{}}} \epsilon_s^{\mathbf{P}} \text{ if } p_d > 0 \\ 0 \text{ otherwise} \end{cases}$$

$$\underline{\textit{Input Gate:}} \quad \delta_\iota^{\mathbf{P}} = f_1'(a_\iota^{\mathbf{P}}) f_2(a_c^{\mathbf{P}}) \epsilon_s^{\mathbf{P}}$$

## 2.2 Connectionist Temporal Classification

Connectionist temporal classification (CTC) [5] is an output layer designed for sequence labelling with RNNs. Unlike other neural network output layers it does not require pre-segmented training data, or postprocessing to transform its outputs into transcriptions. Instead, it trains the network to directly estimate the conditional probabilities of the possible labellings given the input sequences.

A CTC output layer contains one more unit than there are elements in the alphabet $L$ of labels for the task. The output activations are normalised at each timestep with the softmax activation function [2]. The first $|L|$ outputs estimate the probabilities of observing the corresponding labels at that time, and the extra output estimates the probability of observing a 'blank', or no label. The combined output sequence estimates the joint probability of all possible alignments of the input sequence with all sequences of labels and blanks. The probability of a particular labelling can then be estimated by summing over the probabilities of all the alignments that correspond to it.

More precisely, for a length $T$ input sequence $\mathbf{x}$, the CTC outputs define a probability distribution over the set $L'^{T}$ of length T sequences over the alphabet $L' = L \cup \{blank\}$. To distinguish them from labellings, we refer to the elements of $L'^{T}$ as *paths*. Since the probabilities of the labels at each timestep are conditionally independent given $\mathbf{x}$, the conditional probability of a path $\pi \in L'^{T}$ is given by $p(\pi|\mathbf{x}) = \prod_{t=1}^{T} y_{\pi_t}^{t}$. where $y_k^t$ is the activation of output unit $k$ at time $t$.

Paths are mapped onto labellings $\mathbf{l} \in \mathbf{L}^{\leq T}$ by an operator $\mathcal{B}$ that removes first the repeated labels, then the blanks. So for example, both $\mathcal{B}(a, -, a, b, -)$ and $\mathcal{B}(-, a, a, -, -, a, b, b)$ yield the labelling $(a, a, b)$. Since the paths are mutually exclusive, the conditional probability of some labelling $\mathbf{l} \in \mathbf{L}^{\leq T}$ is the sum of the probabilities of all paths corresponding to it: $p(\mathbf{l}|\mathbf{x}) = \sum_{\pi \in \mathcal{B}^{-1}(\mathbf{l})} p(\pi|\mathbf{x})$. Although a naive calculation of this sum is unfeasible, it can be efficiently evaluated with a dynamic programming algorithm, similar to the forward-backward algorithm for HMMs.

To allow for blanks in the output paths, for each labelling $\mathbf{l} \in \mathbf{L}^{\leq T}$ consider a modified labelling $\mathbf{l}' \in \mathbf{L}'^{\leq T}$, with blanks added to the beginning and the end and inserted between every pair of labels. The length $|\mathbf{l}'|$ of $\mathbf{l}'$ is therefore $2|\mathbf{l}| + 1$.

For a labelling $\mathbf{l}$, define the *forward variable* $\alpha_t(s)$ as the summed probability of all path beginnings reaching index $s$ of $\mathbf{l}'$ at time $t$, and the *backward variables* $\beta_t(s)$ as the summed probability of all path endings that would complete the labelling $\mathbf{l}$ if the path beginning had reached $s$ at time $t$. Both

the forward and backward variables are calculated recursively [5]. The label sequence probability is given by the sum of the products of the forward and backward variables at any timestep, i.e. $p(\mathbf{l}|\mathbf{x}) = \sum_{s=1}^{|\mathbf{l}'|} \alpha_t(s)\beta_t(s)$.

Let $S$ be a training set, consisting of pairs of input and target sequences $(\mathbf{x}, \mathbf{z})$, where $|\mathbf{z}| \leq |\mathbf{x}|$. Then the objective function $O$ for CTC is the negative log probability of the network correctly labelling all of S: $O = -\sum_{(\mathbf{x},\mathbf{z})\in S} \ln p(\mathbf{z}|\mathbf{x})$. The network can be trained with gradient descent by first differentiating $O$ with respect to the outputs, then using backpropagation through time to find the derivatives with respect to the weights.

Note that the same label (or blank) may be repeated several times for a single labelling $\mathbf{l}$. We define the set of positions where label $k$ occurs as $lab(\mathbf{l}, k) = \{s : \mathbf{l}'_s = k\}$, which may be empty. Setting $\mathbf{l} = \mathbf{z}$ and differentiating $O$ with respect to the network outputs, we obtain:

$$-\frac{\partial O}{\partial a_k^t} = -\frac{\partial \ln p(\mathbf{z}|\mathbf{x})}{\partial a_k^t} = y_k^t - \frac{1}{p(\mathbf{z}|\mathbf{x})} \sum_{s \in lab(\mathbf{z},k)} \alpha_t(s)\beta_t(s),$$

where $a_k^t$ and $y_k^t$ are respectively the input and output of CTC unit $k$ at time $t$ for some $(\mathbf{x}, \mathbf{z}) \in S$.

Once the network is trained, we can label some unknown input sequence $\mathbf{x}$ by choosing the labelling $\mathbf{l}^*$ with the highest conditional probability, i.e. $\mathbf{l}^* = \arg\max_{\mathbf{l}} p(\mathbf{l}|\mathbf{x})$. In cases where a dictionary is used, the labelling can be constrained to yield only sequences of complete words by using the CTC token passing algorithm [6]. For the experiments in this paper, the labellings were further constrained to give single word sequences only, and the ten most probable words were recorded.

## 2.3  Network Hierarchy

Many computer vision systems use a hierarchical approach to feature extraction, with the features at each level used as input to the next level [15]. This allows complex visual properties to be built up in stages. Typically, such systems use subsampling, with the feature resolution decreased at each stage. They also generally have more features at the higher levels. The basic idea is to progress from a small number of simple local features to a large number of complex global features.

We created a hierarchical structure by repeatedly composing MDLSTM layers with feedforward layers. The basic procedure is as follows: (1) the image is divided into small pixel blocks, each of which is presented as a single input to the first set of MDLSTM layers (e.g. a 4x3 block is reduced to a length 12 vector). If the image does not divide exactly into blocks, it is padded with zeros. (2) the four MDLSTM layers scan through the pixel blocks in all directions. (3) the activations of the MDLSTM layers are collected into blocks. (4) these blocks are given as input to a feedforward layer. Note that all the layers have a 2D array of activations: e.g. a 10 unit feedforward layer with input from a 5x5 array of MDLSTM blocks has a total of 250 activations.

The above process is repeated as many times as required, with the activations of the feedforward layer taking the place of the original image. The purpose of the blocks is twofold: to collect local contextual information, and to reduce the area of the activation arrays. In particular, we want to reduce the vertical dimension, since the CTC output layer requires a 1D sequence as input. Note that the blocks themselves do not reduce the overall amount of data; that is done by the layers that process them, which are therefore analogous to the subsampling steps in other approaches (although with trainable weights rather than a fixed subsampling function).

For most tasks we find that a hierarchy of three MDLSTM/feedforward stages gives the best results. We use the standard 'inverted pyramid' structure, with small layers at the bottom and large layers at the top. As well as allowing for more features at higher levels, this leads to efficient networks, since most of the weights are concentrated in the upper layers, which have a smaller input area.

In general we cannot assume that the input images are of fixed size. Therefore it is difficult to choose block heights that ensure that the final activation array will always be one-dimensional, as required by CTC. A simple solution is to collapse the final array by summing over all the inputs in each vertical line, i.e. the input at time $t$ to CTC unit $k$ is given by $a_k^t = \sum_x a_k^{(x,t)}$, where $a_k^{(x,y)}$ is the uncollapsed input to unit $k$ at point $(x, y)$ in the final array.

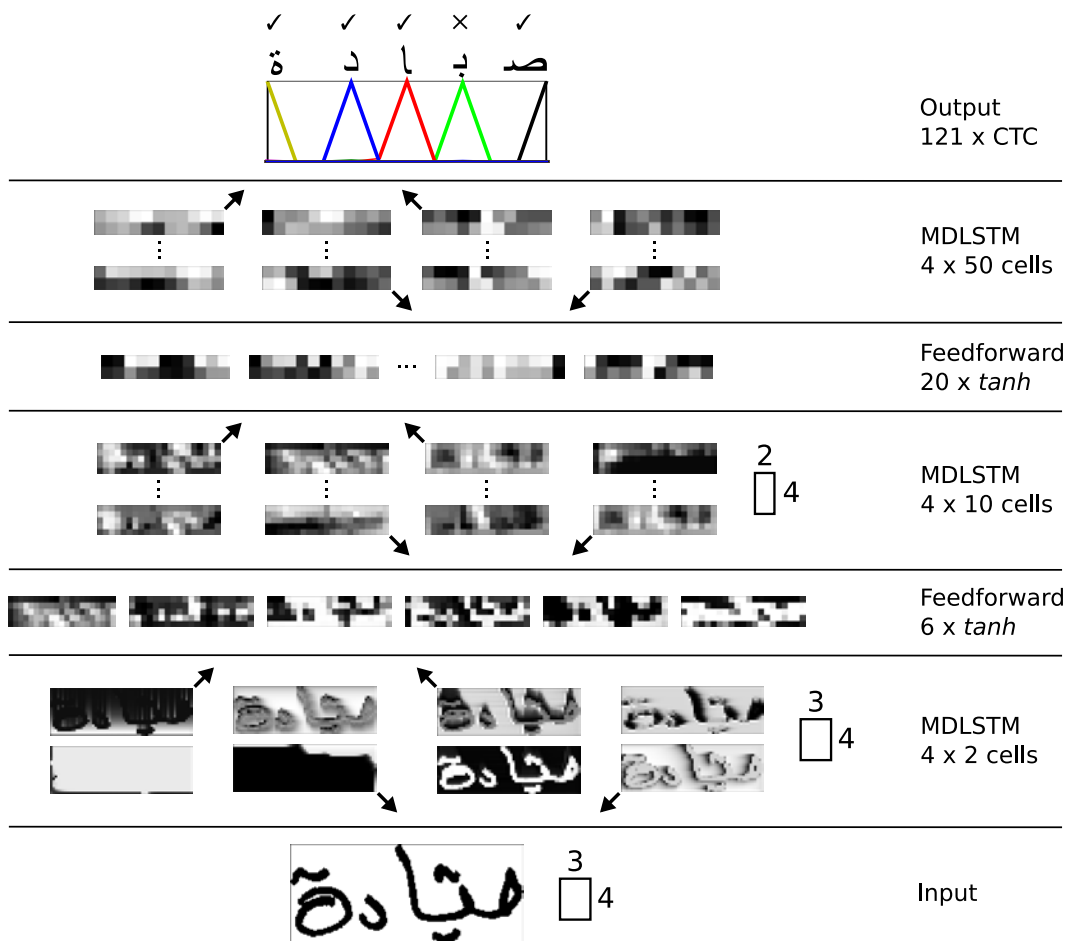

Figure 2: **The complete recognition system**. First the input image is collected into boxes 3 pixels wide and 4 pixels high which are then scanned by four MDLSTM layers. The activations of the cells in each layer are displayed separately, and the arrows in the corners indicates the scanning direction. Next the MDLSTM activations are gathered into 4 x 3 boxes and fed to a feedforward layer of $tanh$ summation units. This process is repeated two more times, until the final MDLSTM activations are collapsed to a 1D sequence and transcribed by the CTC layer. In this case all characters are correctly labelled except the second last one, and the correct town name is chosen from the dictionary.

## 3 Experiments

To see how our method compared to the state of the art, we applied it to data from the ICDAR 2007 Arabic handwriting recognition competition [12]. Although we were too late to enter the competition itself, the organisers kindly agreed to evaluate our system according to the competition criteria. We did not receive the test data at any point, and all evaluations were carried out by them.

The goal of the competition was to identify the postcodes of Tunisian town and village names. The names are presented individually, so it is an isolated word recognition task. However we would like to point out that our system is equally applicable to unconstrained handwriting, and has been successfully applied to complete lines of English text.

### 3.1 Data

The competition was based on the IFN/ENIT database of handwritten Arabic words [13]. The publically available data consists of 32,492 images of handwritten Tunisian town names, of which we used 30,000 for training, and 2,492 for validation. The images were extracted from artificial

Table 1: **Results on the ICDAR 2007 Arabic handwriting recognition contest**. All scores are percentages of correctly identified postcodes. The systems are ordered by the 'top 1' results on test set 'f'. The best score in each column is shown in bold.

| | SET f | | | SET s | | |
|---|---|---|---|---|---|---|
| SYSTEM | top 1 | top 5 | top 10 | top 1 | top 5 | top 10 |
| CACI-3 | 14.28 | 29.88 | 37.91 | 10.68 | 21.74 | 30.20 |
| CACI-2 | 15.79 | 21.34 | 22.33 | 14.24 | 19.39 | 20.53 |
| CEDAR | 59.01 | 78.76 | 83.70 | 41.32 | 61.98 | 69.87 |
| MITRE | 61.70 | 81.61 | 85.69 | 49.91 | 70.50 | 76.48 |
| UOB-ENST-1 | 79.10 | 87.69 | 90.21 | 64.97 | 78.39 | 82.20 |
| PARIS V | 80.18 | 91.09 | 92.98 | 64.38 | 78.12 | 82.13 |
| ICRA | 81.47 | 90.07 | 92.15 | 72.22 | 82.84 | 86.27 |
| UOB-ENST-2 | 81.65 | 90.81 | 92.35 | 69.61 | 83.79 | 85.89 |
| UOB-ENST-4 | 81.81 | 88.71 | 90.40 | 70.57 | 79.85 | 83.34 |
| UOB-ENST-3 | 81.93 | 91.20 | 92.76 | 69.93 | 84.11 | 87.03 |
| SIEMENS-1 | 82.77 | 92.37 | 93.92 | 68.09 | 81.70 | 85.19 |
| MIE | 83.34 | 91.67 | 93.48 | 68.40 | 80.93 | 83.73 |
| SIEMENS-2 | 87.22 | 94.05 | 95.42 | 73.94 | 85.44 | 88.18 |
| Ours | **91.43** | **96.12** | **96.75** | **78.83** | **88.00** | **91.05** |

forms filled in by over 400 Tunisian people. The forms were designed to simulate writing on a letter, and contained no lines or boxes to constrain the writing style.

Each image was supplied with a ground truth transcription for the individual characters[1]. There were 120 distinct characters in total. A list of 937 town names and postcodes was provided. Many of the town names had transcription variants, giving a total of 1,518 entries in the complete dictionary.

The test data (which is not published) was divided into sets 'f' and 's'. The main competition results were based on set 'f'. Set 's' contains data collected in the United Arab Emirates using the same forms; its purpose was to test the robustness of the recognisers to regional writing variations. The systems were allowed to choose up to 10 postcodes for each image, in order of preference. The test set performance using the top 1, top 5, and top 10 answers was recorded by the organisers.

## 3.2 Network Parameters

The structure shown in Figure 2 was used, with each layer fully connected to the next layer in the hierarchy, all MDLSTM layers connected to themselves, and all units connected to a bias weight. There were 159,369 weights in total. This may sound like a lot, but as mentioned in Section 2.3, the 'inverted pyramid' structure greatly reduces the actual number of weight operations. In effect the higher up networks (where the vast majority of the weights are concentrated) are processing much smaller images than those lower down. The squashing function for the gates was the logistic sigmoid $f_1(x) = 1/(1 + e^{-x})$, while $tanh$ was used for $f_2$ and $f_3$. Each pass through the training set took about an hour on a desktop computer, and the network converged after 85 passes.

The complete system was trained with online gradient descent, using a learning rate of $10^{-4}$ and a momentum of $0.9$. The character error rate was evaluated on the validation set after every pass through the training set, and training was stopped after 50 evaluations with no improvement. The weights giving the lowest error rate on the validation set were passed to the competition organisers for assessment on the test sets.

## 3.3 Results

Table 1 clearly shows that our system outperformed all entries in the 2007 ICDAR Arabic recognition contest. The other systems, most of which are based on hidden Markov models, are identified by the names of the groups that submitted them (see [12] for more information).

# 4   Conclusions and Future Work

We have combined multidimensional LSTM with connectionist temporal classification and a hierarchical layer structure to create a powerful offline handwriting recogniser. The system is very general, and has been successfully applied to English as well as Arabic. Indeed, since the dimensionality of the networks can be changed to match that of the data, it could in principle be used for almost any supervised sequence labelling task.

**Acknowledgements**

We would like to thank Haikal El Abed for giving us access to the ICDAR competition data, and for persisting in the face of technical despair to install and evaluate our software. This work was supported by the excellence cluster "Cognition for Technical Systems" (CoTeSys) from the German Research Foundation (DFG).

## Footnotes

[1]At first we forgot that Arabic reads right to left and presented the transcriptions backwards. The system performed surprisingly well, with a character error rate of 17.8%, compared to 10.7% for the correct targets.

# References

[1] P. Baldi and G. Pollastri. The principled design of large-scale recursive neural network architectures–dag-rnns and the protein structure prediction problem. *J. Mach. Learn. Res.*, 4:575–602, 2003.

[2] J. S. Bridle. Probabilistic interpretation of feedforward classification network outputs, with relationships to statistical pattern recognition. In F. Fogleman-Soulie and J.Herault, editors, *Neurocomputing: Algorithms, Architectures and Applications*, pages 227–236. Springer-Verlag, 1990.

[3] F. Gers, N. Schraudolph, and J. Schmidhuber. Learning precise timing with LSTM recurrent networks. *Journal of Machine Learning Research*, 3:115–143, 2002.

[4] A. Graves. *Supervised Sequence Labelling with Recurrent Neural Networks*. PhD thesis.

[5] A. Graves, S. Fernández, F. Gomez, and J. Schmidhuber. Connectionist temporal classification: Labelling unsegmented sequence data with recurrent neural networks. In *Proceedings of the International Conference on Machine Learning, ICML 2006*, Pittsburgh, USA, 2006.

[6] A. Graves, S. Fernández, M. Liwicki, H. Bunke, and J. Schmidhuber. Unconstrained online handwriting recognition with recurrent neural networks. In J. Platt, D. Koller, Y. Singer, and S. Roweis, editors, *Advances in Neural Information Processing Systems 20*. MIT Press, Cambridge, MA, 2008.

[7] A. Graves, S. Fernández, and J. Schmidhuber. Multidimensional recurrent neural networks. In *Proceedings of the 2007 International Conference on Artificial Neural Networks*, Porto, Portugal, September 2007.

[8] S. Hochreiter and J. Schmidhuber. Long Short-Term Memory. *Neural Computation*, 9(8):1735–1780, 1997.

[9] J. Hu, S. G. Lim, and M. K. Brown. Writer independent on-line handwriting recognition using an HMM approach. *Pattern Recognition*, 33:133–147, 2000.

[10] S. Jaeger, S. Manke, J. Reichert, and A. Waibel. On-line handwriting recognition: the NPen++ recognizer. *International Journal on Document Analysis and Recognition*, 3:169–180, 2001.

[11] Y. LeCun, L. Bottou, Y. Bengio, and P. Haffner. Gradient-based learning applied to document recognition. *Proceedings of the IEEE*, 86(11):2278–2324, November 1998.

[12] V. Margner and H. E. Abed. Arabic handwriting recognition competition. In *ICDAR '07: Proceedings of the Ninth International Conference on Document Analysis and Recognition (ICDAR 2007) Vol 2*, pages 1274–1278, Washington, DC, USA, 2007. IEEE Computer Society.

[13] M. Pechwitz, S. S. Maddouri, V. Mrgner, N. Ellouze, and H. Amiri. IFN/ENIT-database of handwritten arabic words. In *7th Colloque International Francophone sur l'Ecrit et le Document (CIFED 2002)*, Hammamet, Tunis, 2002.

[14] R. Plamondon and S. N. Srihari. On-line and off-line handwriting recognition: a comprehensive survey. *IEEE Transactions on Pattern Analysis and Machine Intelligence*, 2000.

[15] M. Reisenhuber and T. Poggio. Hierarchical models of object recognition in cortex. *Nature Neuroscience*, 2(11):1019–1025, 1999.

[16] M. Schuster and K. K. Paliwal. Bidirectional recurrent neural networks. *IEEE Transactions on Signal Processing*, 45:2673–2681, November 1997.

